# Message-Passing for Approximate MAP Inference with Latent Variables

**Jiarong Jiang**
Dept. of Computer Science
University of Maryland, CP
jiarong@umiacs.umd.edu

**Piyush Rai**
School of Computing
University of Utah
piyush@cs.utah.edu

**Hal Daumé III**
Dept. of Computer Science
University of Maryland, CP
hal@umiacs.umd.edu

## Abstract

We consider a general inference setting for discrete probabilistic graphical models where we seek maximum a posteriori (MAP) estimates for a subset of the random variables (max nodes), marginalizing over the rest (sum nodes). We present a hybrid message-passing algorithm to accomplish this. The hybrid algorithm passes a mix of sum and max messages depending on the type of source node (sum or max). We derive our algorithm by showing that it falls out as the solution of a particular relaxation of a variational framework. We further show that the Expectation Maximization algorithm can be seen as an approximation to our algorithm. Experimental results on synthetic and real-world datasets, against several baselines, demonstrate the efficacy of our proposed algorithm.

## 1 Introduction

Probabilistic graphical models provide a compact and principled representation for capturing complex statistical dependencies among a set of random variables. In this paper, we consider the general maximum *a posteriori* (MAP) problem in which we want to maximize over a subset of the variables (max nodes, denoted $X$), marginalizing the rest (sum nodes, denoted $Z$). This problem is termed as the Marginal-MAP problem. A typical example is the minimum Bayes risk (MBR) problem [1] where the goal is to find an assignment $\hat{x}$ which optimizes a loss $\ell(\hat{x}, x)$ with regard to some usually unknown truth $x$. Since $x$ is *latent*, we need to marginalize it before optimizing with respect to $\hat{x}$.

Although the specific problems of estimating marginals and estimating MAP individually have been studied extensively [2, 3, 4], similar developments for the more general problem of simultaneous marginal *and* MAP estimation are lacking. More recently, [5] proposed a method based optimizing a variational objective on specific graph structures, and is a simultaneous development as the method we propose in this paper (please refer to the supplementary material for further details and other related work).

This problem is fundamentally difficult. As mentioned in [6, 7], even for a tree-structured model, we cannot solve the Marginal-MAP problem exactly in poly-time unless $P = NP$. Moreover, it has been shown [8] that even if a joint distribution $p(x, z)$ belongs to the exponential family, the corresponding marginal distribution $p(x) = \sum_z p(x, z)$ is in general *not* exponential family (with a very short list of exceptions, such as Gaussian random fields). This means that we cannot directly apply algorithms for MAP inference to our task. Motivated by this problem, we propose a hybrid message passing algorithm which is both intuitive and justified according to variational principles. Our hybrid message passing algorithm uses a mix of sum and max messages with the message type depending on the source node type.

Experimental results on chain and grid structured synthetic data sets and another real-world dataset show that our hybrid message-passing algorithm works favorably compared to standard sum-product, standard max-product, or the Expectation-Maximization algorithm which iteratively provides MAP and marginal estimates. Our estimates can be further improved by a few steps of local

search [6]. Therefore, using the solution found by our hybrid algorithm to initialize some local search algorithms largely improves the performance on both accuracy and convergence speed, compared to the greedy stochastic search method described in [6]. We also give an example in Sec. 5 of how our algorithm can also be used to solve other practical problem which can be cast under the Marginal-MAP framework. In particular, the Minimum Bayes Risk [9] problem for decomposable loss-functions can be readily solved under this framework.

## 2 Problem Setting

In our setting, the nodes in a graphical model with discrete random variables are divided into two sets: *max* and *sum* nodes. We denote a graph $G = (V, E)$, $V = X \cup Z$ where $X$ is the set of nodes for which we want to compute the MAP assignment (*max nodes*), and $Z$ is the set of nodes for which we need the marginals (*sum nodes*). Let $x = \{x_1, \ldots, x_m\}$ ($x_s \in \mathcal{X}_s$), $z = \{z_1, \ldots, z_n\}$ ($z_s \in \mathcal{Z}_s$) be the random variables associated with the nodes in $X$ and $Z$ respectively. The exponential family distribution $p$ over these random variables is defined as follows:

$$p_\theta(x, z) = \exp\left[\langle \theta, \phi(x, z) \rangle - A(\theta)\right]$$

where $\phi(x, z)$ is the sufficient statistics of the enumeration of all node assignments, and $\theta$ is the vector of canonical or exponential parameters. $A(\theta) = \log \sum_{x,z} \exp[\langle \theta, \phi(x, z) \rangle]$ is the log-partition function. In this paper, we consider only pairwise node interactions and use standard overcomplete representation of the sufficient statistics [10] (defined by indicator function $\mathbb{I}$ later).

The general MAP problem can be formalized as the following maximization problem:

$$x^* = \arg \max_x \sum_z p_\theta(x, z) \tag{1}$$

with corresponding marginal probabilities of the $z$ nodes, given $x^*$.

$$p(z_s|x^*) = \sum_{Z \backslash \{z_s\}} p(z|x^*), \quad s = 1, \ldots, n \tag{2}$$

Before proceeding, we introduce some notations for clarity of exposition: Subscripts $s$, $u$, $t$, etc. denote nodes in the graphical model. $z_s$, $x_s$ are sum and max *random variables* respectively, associated with some node $s$. $v_s$ can be either a sum ($z_s$) or a max random ($x_s$) variable, associated with some node $s$. $N(s)$ is the set of neighbors of node $s$. $\mathcal{X}_s, \mathcal{Z}_s, \mathcal{V}_s$ are the state spaces from which $x_s$, $z_s$, $v_s$ take values.

### 2.1 Message Passing Algorithms

The sum-product and max-product algorithms are standard message-passing algorithms for inferring marginal and MAP estimates respectively in probabilistic graphical models. Their idea is to store a *belief state* associated with each node, and iteratively passing messages between adjacent nodes, which are used to update the belief states. It is known [11] that these algorithms are guaranteed to converge to the exact solution on trees or polytrees. On loopy graphs, they are no longer guaranteed to converge, but they can still provide good estimates when converged [12].

In the standard sum product algorithm, the message $M_{ts}$ passed from node $s$ to one of its neighbors $t$ is as follows:

$$M_{ts}(v_s) \leftarrow \kappa \sum_{v_t' \in \mathcal{V}_t} \left\{ \exp[\theta_{st}(v_s, v_t') + \theta_t(v_t')] \prod_{u \in N(t) \backslash s} M_{ut}(v_t') \right\} \tag{3}$$

where $\kappa$ is a normalization constant. When the messages converge, i.e. $\{M_{ts}, M_{st}\}$ does not change for every pair of nodes $s$ and $t$, the belief (psuedomarginal distribution) for the node $s$ is given by $\mu_s(v_s) = \kappa \exp\{\theta_s(v_s)\} \prod_{t \in N(s)} M_{ts}(v_s)$. The outgoing messages for max product algorithm have the same form but with a maximization instead of a summation in Eq. (3). After convergence, the MAP assignment for each node is the assignment with the highest max-marginal probability.

On loopy graphs, the tree-weighted sum and max product [13, 14] can help find the upper bound of the marginal or MAP problem. They decompose the loopy graph into several spanning trees and reweight the messages by the edge appearance probability.

## 2.2 Local Search Algorithm

Eq (1) can be viewed as doing a variable elimination for $z$ nodes first, followed by a maximization over $x$. Its maximization step may be performed using heuristic search techniques [7, 6]. Eq (2) can be computed by running standard sum-product over $z$, given the MAP $x^*$ assignments. In [6], the assignment for the MAP nodes are found by greedily searching the best neighboring assignments which only differs on one node. However, the hybrid algorithm we propose allows simultaneously approximating both Eq (1) and Eq (2).

## 3 HYBRID MESSAGE PASSING

In our setting, we wish to compute MAP estimates for one set of nodes and marginals for the rest. One possible approach is to run standard sum/max product algorithms over the graph, and find the most-likely assignment for each max node according to the maximum of sum or max marginals[1]. These naïve approaches have their own shortcomings; for example, although using standard max-product may perform reasonably when there are many max nodes, it inevitably ignores the effect of sum nodes which should ideally be summed over. This is analogous to the difference between EM for Gaussian mixture models and $K$-means. (See Sec. 6)

## 3.1 ALGORITHM

We now present a hybrid message-passing algorithm which passes sum-style or max-style messages based on the type of nodes from which the message originates. In the hybrid message-passing algorithm, a sum node sends sum messages to its neighbors and a max node sends max messages. The type of message passed depends on the type of source node, not the destination node.

More specifically, the outgoing messages from a source node are as follows:
- Message from *sum* node $t$ to any neighbor $s$:

$$M_{ts}(v_s) \leftarrow \kappa_1 \sum_{z_t' \in \mathcal{Z}_t} \left\{ \exp[\theta_{st}(v_s, z_t') + \theta_t(z_t')] \prod_{u \in N(t) \backslash s} M_{ut}(z_t') \right\} \tag{4}$$

- Message from *max* node $t$ to any neighbor $s$:

$$M_{ts}(v_s) \leftarrow \kappa_2 \max_{x_t' \in \mathcal{X}_t} \left\{ \exp[\theta_{st}(v_s, x_t') + \theta_t(x_t')] \prod_{u \in N(t) \backslash s} M_{ut}(x_t') \right\} \tag{5}$$

and $\kappa_1, \kappa_2$ are normalization constants. Algo 1 shows the procedure to do hybrid message-passing.

---

**Algorithm 1** Hybrid Message-Passing Algorithm

---

**Inputs:** Graph $G = (V, E)$, $V = X \cup Z$, potentials $\theta_s$, $s \in V$ and $\theta_{st}$, $(s, t) \in E$.
    1. Initialize the messages to some arbitrary value.
    2. For each node $s \in V$ in $G$, do the following until messages converge (or maximum number of iterations reached)
        • If $s \in X$, update messages by Eq.(5).
        • If $s \in Z$, update messages by Eq.(4).
    3. Compute the local belief for each node $s$.
        $\mu_s(y_s) = \kappa \exp\{\theta_s(v_s)\} \prod_{t \in N(s)} M_{ts}(v_s)$
    4. For all $x_s \in X$, return $\arg\max_{x_s \in \mathcal{X}_s} \mu_s(x_s)$
    5. For all $z_s \in Z$, return $\mu_s(z_s)$.

---

When there is only a single type of node in the graph, the hybrid algorithm reduces to the standard max or sum-product algorithm. Otherwise, it passes different messages simultaneously and gives an approximation to the MAP assignment on max nodes as well as the marginals on sum nodes. On the loopy graphs, we can also apply this scheme to pass hybrid tree-reweighted messages between nodes to obtain marginal and MAP estimates. (See Appendix C of the supplementary material)

## 3.2 VARIATIONAL DERIVATION

In this section, we show that the Marginal-MAP problem can be framed under a variational framework, and the hybrid message passing algorithm turns out to be a solution of it. (a detailed derivation is in Appendix A of the supplementary material). To see this, we construct a new graph $G_{\bar{\mathbf{x}}}$ with $x$s' assignments fixed to be $\bar{\mathbf{x}} \in \mathbf{X} = \mathcal{X}_1 \times \cdots \times \mathcal{X}_m$, so the log-partition function $A(\theta_{\bar{\mathbf{x}}})$ of the graph $G_{\bar{\mathbf{x}}}$ is

$$A(\theta_{\bar{\mathbf{x}}}) = \log \sum_z p(\bar{\mathbf{x}}, z) + \log A(\theta) = \log p(\bar{\mathbf{x}}) + const \tag{6}$$

As the constant only depends on the log-partition function of the original graph and does not vary with different assignments of MAP nodes, $A(\theta_{\bar{\mathbf{x}}})$ exactly estimates the log-likelihood of assignment $\bar{\mathbf{x}}$. Therefore $\text{argmax}_{\bar{\mathbf{x}} \in \mathcal{X}} \log p(\bar{\mathbf{x}}) = \text{argmax}_{\bar{\mathbf{x}} \in \mathcal{X}} A(\theta_{\bar{\mathbf{x}}})$. Moreover, $A(\theta_{\bar{\mathbf{x}}})$ can be approximated by the following [10]:

$$A(\theta_{\bar{\mathbf{x}}}) \approx \sup_{\mu \in M(G_{\bar{\mathbf{x}}})} \langle \theta, \mu \rangle + H_{Bethe}(\mu) \tag{7}$$

where $M(G_{\bar{\mathbf{x}}})$ is the following marginal polytope of graph $G_{\mathbf{x}}$:

$$M(G_{\bar{x}}) = \left\{ \mu \;\middle|\; \begin{array}{l} \mu_s(z_s), \mu_{st}(v_s, v_t): \text{marginals with } \bar{\mathbf{x}} \text{ fixed to its assignment} \\ \mu_s(x_s) = \left\{ \begin{array}{ll} 1 & \text{if } x_s = \bar{x}_s \\ 0 & \text{else} \end{array} \right. \end{array} \right\} \tag{8}$$

Recall, $v_s$ stands for $x_s$ or $z_s$. $H_{Bethe}(\mu)$ is the Bethe energy of the graph:

$$H_{Bethe}(\mu) = \sum_s H_s(\mu_s) - \sum_{(s,t) \in E} I_{st}(\mu_{st}), H_s(\mu_s) = - \sum_{v_s \in \mathcal{V}_s} \mu_s(v_s) \log \mu_s(v_s)$$

$$I_{st}(\mu_{st}) = \sum_{(v_s, v_t) \in \mathcal{V}_s \times \mathcal{V}_t} \mu_{st}(v_s, v_t) \log \frac{\mu_{st}(v_s, v_t)}{\mu_s(v_s)\mu_t(v_t)} \tag{9}$$

For readability, we use $\mu_{\text{sum}}, \mu_{\text{max}}$ to subsume the node and pairwise marginals for sum/max nodes and $\mu_{\text{sum} \rightarrow \text{max}}, \mu_{\text{max} \rightarrow \text{sum}}$ are the pairwise marginals for edges between different types of nodes. The direction here is used to be consistent with the distinction of the constraints as well as the messages.

Solving the Marginal-MAP problem is therefore equivalent to solving the following optimization problem:

$$\max_{\bar{\mathbf{x}} \in \mathbf{X}} \sup_{\mu_{\text{other}} \in M(G_{\bar{\mathbf{x}}})} \langle \theta, \mu \rangle + H_{Bethe}(\mu) \approx \sup_{\mu_{\text{max}} \in M_{\bar{\mathbf{x}}}} \sup_{\mu_{\text{other}} \in M(G_{\bar{\mathbf{x}}})} \langle \theta, \mu \rangle + H_{Bethe}(\mu) \tag{10}$$

$\mu_{\text{other}}$ contains all other node/pairwise marginals except $\mu_{\text{max}}$. The Bethe entropy terms can be written as ($H$ is the entropy and $I$ is mutual information)

$$H_{Bethe}(\mu) = H_{\mu_{\text{max}}} + H_{\mu_{\text{sum}}} - I_{\mu_{\text{max}} \rightarrow \mu_{\text{max}}} - I_{\mu_{\text{sum}} \rightarrow \mu_{\text{sum}}} - I_{\mu_{\text{max}} \rightarrow \mu_{\text{sum}}} - I_{\mu_{\text{sum}} \rightarrow \mu_{\text{max}}}$$

If we force to satisfy the second condition in (8), the entropy of max nodes $H_{\mu_{\text{max}}} = H_s(\mu_s) = 0$, $\forall s \in X$ and the mutual information between max nodes $I_{\mu_{\text{max}} \rightarrow \mu_{\text{max}}} = I_{st}(x_s, x_t) = 0, \forall s, t \in X$. For mutual information between different types of nodes, we can either force $x$s to have integral solutions, or relax $x$s to have non-integral solution, or relax $x$s on one direction[2]. In practice, we relax the mutual information on the message from sum nodes to max nodes, so the mutual information on the other direction $I_{\mu_{\text{max}} \rightarrow \mu_{\text{sum}}} = I_{st}(x_s, z_t) = \sum_{(x_s, z_t) \in \mathcal{X}_s \times \mathcal{Z}_t} \mu_{st}(x_s, z_t) \log \frac{\mu_{st}(x_s, z_t)}{\mu_s(x_s)\mu_t(z_t)} = \sum_{z_t \in \mathcal{Z}_t} \mu_{st}(x^*, z_t) \log \frac{\mu_{st}(x^*, z_t)}{\mu_s(x^*)\mu_t(z_t)} = 0, \forall s \in X, t \in Z$, where $x^*$ is the assigned state of $x$ at node $s$. Finally, we only require sum nodes to satisfy normalization and marginalization conditions, the entropy for sum nodes, mutual information between sum nodes, and from sum node to max node can be nonzero.

The above process relaxes the polytope $M(G_{\bar{\mathbf{x}}})$ to be $M_{\bar{\mathbf{x}}} \times L_z(G_{\bar{\mathbf{x}}})$, where

$$L_z(G_{\bar{x}}) = \left\{ \mu \geq 0 \;\middle|\; \begin{array}{l} \sum_{z_s} \mu_s(z_s) = 1, \mu_s(x_s) = 1 \text{ iff } x_s = \bar{x}_s, \\ \sum_{z_t} \mu_{st}(v_s, z_t) = \mu_s(v_s), \\ \sum_{z_s} \mu_{st}(z_s, v_t) = \mu_t(v_t), \\ \mu_{st}(x_s, z_t) = \mu_t(z_t) \text{ iff } x_s = \bar{x}_s, \\ \mu_{st}(x_s, x_t) = 1 \text{ iff } x_s = \bar{x}_s, x_t = \bar{x}_t. \end{array} \right\}$$

This analysis results in the following optimization problem.

$$\sup_{\mu_{\max} \in M_{\bar{x}}} \sup_{\mu_{\text{others}} \in M(G_{\bar{x}})} \langle \theta, \mu \rangle + H(\mu_{\text{sum}}) - I(\mu_{\text{sum} \to \text{sum}}) - I(\mu_{\text{sum} \to \text{max}})$$

Further relaxing $\mu_{\bar{x}}$s to have non-integral solutions, define

$$L(G) = \left\{ \mu \geq 0 \;\middle|\; \begin{array}{l} \sum_{v_s} \mu_s(v_s) = 1, \\ \sum_{v_t} \mu_{st}(v_s, v_t) = \mu_s(v_s), \\ \sum_{v_s} \mu_{st}(v_s, v_t) = \mu_t(v_t). \end{array} \right\}$$

Finally we get

$$\sup_{\mu \in L(G)} \langle \mu, \theta \rangle + H(\mu_{\text{sum}}) - I(\mu_{\text{sum} \to \text{sum}}) - I(\mu_{\text{sum} \to \text{max}}) \tag{11}$$

So $M_{\bar{x}} \times M_z(G_{\bar{x}}) \subseteq M_{\bar{x}} \times L_z(G_{\bar{x}}) \subseteq L(G)$. Unfortunately, $M_{\bar{x}} \times M_z(G_{\bar{x}})$ is not guaranteed to be convex and we can only obtain an approximate solution to the problem defined in Eq (11). Taking the Lagrangian formulation, for an $x$ node, the partial derivative of the Lagrangian with respect to $\mu_s(x_s), s \in X$ keeps the same form as in max product derivation[10], and the situations are identical for $\mu_s(z_s), s \in Z$ and pairwise psuedo-marginals, so the hybrid message-passing algorithm provides a solution to Eq (11) (see Appendix A of the supplementary material for a detailed derivation).

# 4 Expectation Maximization

Another plausible approach to solve the Marginal MAP problem is by the Expectation Maximization(EM) algorithm [17], typically used for maximum likelihood parameter estimation in latent variable models. In our setting, the variables $Z$ correspond to the latent variables. We now show one way of approaching this problem by applying the sum-product and max-product algorithms in the E and M step respectively. To see this, let us first define[3]:

$$F(\tilde{p}, x) = E_{\tilde{p}}[\log p(x, z)] + H(\tilde{p}(z)) \tag{12}$$

where $H(\tilde{p}) = -E_{\tilde{p}}[\log \tilde{p}(z)]$.

Then EM can be interpreted as a joint maximization of the function $F$ [18]: At iteration $t$, for the E-step, $\tilde{p}^{(t)}$ is set to be the $\tilde{p}$ that maximizes $F(\tilde{p}, x^{(t-1)})$ and for the M-step, $x^{(t)}$ is the $x$ that maximizes $F(\tilde{p}^{(t)}, x)$. Given $F$, the following two properties[4] show that jointly maximizing function $F$ is equivalent to maximizing the objective function $p(x) = \sum_z p(x, z)$.

1. With the value of $x$ fixed in function $F$, the unique solution to maximizing $F(\tilde{p}, x)$ is given by $\tilde{p}(z) = p(z|x)$.
2. If $\tilde{p}(z) = p(z|x)$, then $F(\tilde{p}, x) = \log p(x) = \log \sum_z p(x, z)$.

## 4.1 Expectation Maximization via Message Passing

Now we can derive the EM algorithm for solving the Marginal-MAP problem by jointly maximizing function $F$. In the E-step, we need to estimate $\tilde{p}(z) = p(z|x)$ given $x$. This can be done by fixing $x$ values at their MAP assignments and running the sum-product algorithm over the resulting graph:

The M-step works by maximizing $\mathbb{E}_{p_\theta(z \mid \bar{x})} \log p_\theta(x, z)$, where $\bar{x}$ is the assignment given by the previous M-step. This is equivalent to maximizing $\mathbb{E}_{z \sim p_\theta(z \mid \bar{x})} \log p_\theta(x \mid z)$, as the $\log p_\theta(z)$ term in the maximization is independent of $x$. $\max_x \mathbb{E}_{z \sim p_\theta(z \mid \bar{x})} \log p_\theta(x \mid z) = \max_x \sum_z p(z \mid \bar{x}) \langle \theta, \phi(x, z) \rangle$, which in the overcomplete representation [10] can be approximated by

$$\sum_{s \in X, i} \left[ \theta_{s;i} + \sum_{t \in Z, j} \mu_{t;j} \theta_{st;ij} \right] \mathbb{I}_{s;i}(x_s) + \sum_{(s,t) \in E, s, t \in X} \sum_{(i,j)} \theta_{st;ij} \mathbb{I}_{st;ij}(x_s, x_t) + C$$

where $C$ subsumes the terms irrelevant to the maximization over $x$, $\mu_t$ is the psuedo-marginal of node $t$ given $\bar{x}$[5]. Then, the M-step amounts to running the max product algorithm with potentials on $x$ nodes modified according to Eq. (13). Summarizing, the EM algorithm for solving marginal-MAP estimation can be interpreted as follows:

- **E-step**: Fix $x$s to be the MAP assignment value from iteration $(k-1)$ and run sum product to get beliefs on sum nodes $z$s, say $\mu_t, t \in Z$.

- **M-step**: Build a new graph $\tilde{G} = (\tilde{V}, \tilde{E})$ only containing the max nodes. $\tilde{V}=X$ and $\tilde{E} = \{(s,t)|\forall(s,t) \in E, s,t \in X\}$. For each max node $s$ in the graph, set its potential as $\tilde{\theta}_{s;i} = \theta_{s;i} + \sum_j \theta_{st;ij}\mu_{t;j}$, where $t \in Z$ and $(s,t) \in E$. $\tilde{\theta}_{st;ij} = \theta_{st;ij} \ \forall(s,t) \in \tilde{E}$. Run max product over this new graph and update the MAP assignment.

## 4.2   Relationship with the Hybrid Algorithm

Apart from the fact that the hybrid algorithm passes different messages simultaneously and EM does it iteratively, to see the connection with the hybrid algorithm, let us first consider the message passed in the E-step at iteration $k$. $x$s are fixed at the last assignment which maximizes the message at iteration $k-1$, denoted as $x^*$ here. The $M_{ut}^{(k-1)}$ are the messages computed at iteration $k-1$.

$$M_{ts}^{(k)}(z_s) = \kappa_1 \{ \exp[\theta_{st}(z_s, x_t^*) + \theta_t(x_t^*)] \prod_{u \in N(t)\backslash s} M_{ut}^{(k-1)}(x_t^*) \} \tag{13}$$

Now assume there exists an iterative algorithm which, at each iteration, computes the messages used in both steps of the message-passing variant of the EM algorithm, denoted $\tilde{M}_{ts}$. Eq (13) then becomes

$$\tilde{M}_{ts}^{(k)}(z_s) == \kappa_1 \max_{x'} \{ \exp[\theta_{st}(z_s, x_t') + \theta_t(x_t')] \prod_{u \in N(t)\backslash s} \tilde{M}_{ut}^{(k-1)}(x_t') \}$$

So the max nodes ($x$'s) should pass the max messages to its neighbors ($z$'s), which is what the hybrid message-passing algorithm does.

In the M-step for EM (as discussed in Sec. 4), all the sum nodes $t$ are removed from the graph and the parameters of the adjacent max nodes are modified as: $\theta_{s;i} = \theta_{s;i} + \sum_j \theta_{st;ij}\mu_{t;j}$. $\mu_t$ is computed by the sum product at the E-step of iteration $k$, and these sum messages are used (in form of the marginals $\mu_t$) in the subsequent M-step (with the sum nodes removed). However, a max node may prefer different assignments according to different neighboring nodes. With such uncertainties, especially during the first a few iterations, it is very likely that making hard decisions will directly lead to the bad local optima. In comparison, the hybrid message passing algorithm passes mixed messages instead of making deterministic assignments in each iteration.

## 5   MBR Decoding

Most work on finding "best" solutions in graphical models focuses on the MAP estimation problem: find the $x$ that maximizes $p_\theta(x)$. In many practical applications, one wishes to find an $x$ that *minimizes* some risk, parameterized by a given loss function. This is the minimum Bayes risk (MBR) setting, which has proven useful in a number of domains, such as speech recognition [9], natural language parsing [19, 20], and machine translation [1]. We are given a loss function $\ell(x, \hat{x})$ which measures the *loss* of $\hat{x}$ assuming $x$ is the truth. We assume losses are non-negative. Given this loss function, the minimum Bayes risk solution is the minimizer of Eq (14):

$$\text{MBR}_\theta = \arg\min_{\hat{x}} \mathbb{E}_{x \sim p}[\ell(x, \hat{x})] = \arg\min_{\hat{x}} \sum_x p(x)\ell(x, \hat{x}) \tag{14}$$

We now assume that $\ell$ decomposes over the structure of $x$. In particular, suppose that: $\ell(x, \hat{x}) = \sum_{c \in \mathcal{C}} \ell(x_c, \hat{x}_c)$, where $\mathcal{C}$ is some set of cliques in $x$, and $x_c$ denotes the variables associated with that clique. For example, for Hamming loss, the cliques are simply the set of pairs of vertices of the form $(x_i, \hat{x}_i)$, and the loss simply counts the number of disagreements. Such decompositionality is widely assumed in structured prediction algorithms [21, 22].

Assume $l_c(x, x') \leq L \ \forall c, x, x'$. Therefore $l(x, x') \leq |C|L$. We can then expand Eq (14) into the following:

$$\text{MBR}_\theta = \arg\min_{\hat{x}} \sum_x p(x)\ell(x, \hat{x}) = \arg\max_{x'} \sum_x p(x)(|C|L - l(x, x'))$$

$$= \arg\max_{\hat{x}} \sum_x \exp\left[ \langle \theta, x \rangle + \log \sum_c [L - \ell(x_c, \hat{x}_c)] - A(\theta) \right]$$

This resulting expression has exactly the same form as the MAP-with-marginal problem, where $x$ is the variable being marginalized and $\hat{x}$ being the variable being maximized. Fig. 1 shows a simple example of transforming a MAP lattice problem into an MBR problem under Hamming loss. Therefore, we can apply our hybrid algorithm to solve the MBR problem.

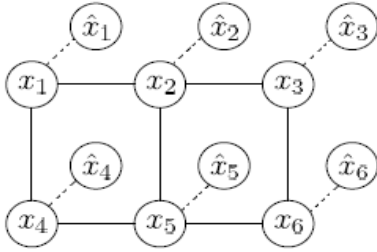

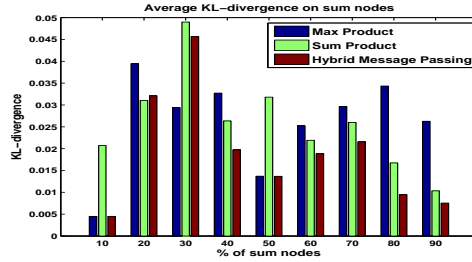

Figure 1: The Augmented Model For Solving The MBR Problem Under Hamming Loss over a 6-node simple lattice

Figure 2: Comparison of Various Algorithms For Marginals on 10-Node Chain Graph

## 6  EXPERIMENTS

We perform the experiments on synthetic datasets as well as a real-world protein side-chain prediction dataset [23], and compare our hybrid message-passing algorithm (both its standard belief propagation and the tree-reweighted belief propagation (TRBP) versions) against a number of baselines such as the standard sum/max product based MAP estimates, EM, TRBP, and the greedy local search algorithm proposed in [6].

### 6.1  Synthetic Data

For synthetic data, we first take a 10-node chain graph with varying splits of sum vs max nodes, and random potentials. Each node can take one of the two states (0/1). The node and the edge potentials are drawn from UNIFORM(0,1) and we randomly pick nodes in the graph to be sum or max nodes. For this small graph, the true assignment is computable by explicitly maximizing $p(x) = \sum_z p(x,z) = \frac{1}{Z} \sum_z \prod_{s \in V} \psi_s(v_s) \prod_{(s,t) \in E} \psi_{st}(v_s, v_t)$, where $Z$ is some normalization constant and $\psi_s(v_s) = \exp \theta_s(v_s)$.

First, we compare the various algorithms on the MAP assignments. Assume that the aforementioned maximization gives assignment $\mathbf{x}^* = (x_1^*, \ldots, x_n^*)$ and some algorithm gives the approximate assignment $\mathbf{x} = (x_1, \ldots, x_n)$. The metrics we use here are the 0/1 loss and the Hamming loss.

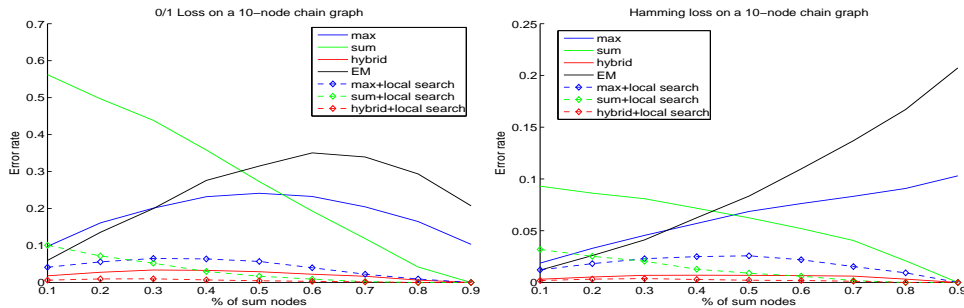

Figure 3: Comparison of Various Algorithms For MAP Estimates on 10-Node Chain Graph: 0-1 Loss (Left), Hamming Loss (Right)

Fig. 3 shows the loss on the assignment of the max nodes. In the figure, as the number of sum nodes goes up, the accuracy of the standard sum-product based estimation (*sum*) gets better, whereas the accuracy of standard max-product based estimation (*max*) worsens. However, our hybrid message-passing algorithm (*hybrid*), on an average, results in the lowest loss compared to the other baselines, with running times similar to the sum/max product algorithms.

We also compare a stochastic greedy search approach described in [6] initialized by the results of sum/max/hybrid algorithm (*sum/max/hybrid+local search*). As shown in [6], local search with sum product initialization empirically performs better than with max product, so later on, we only compare the results with local search using sum product initialization (*LS*). Best of the three initialization methods, by starting at the hybrid algorithm results, the search algorithm in very few steps can find

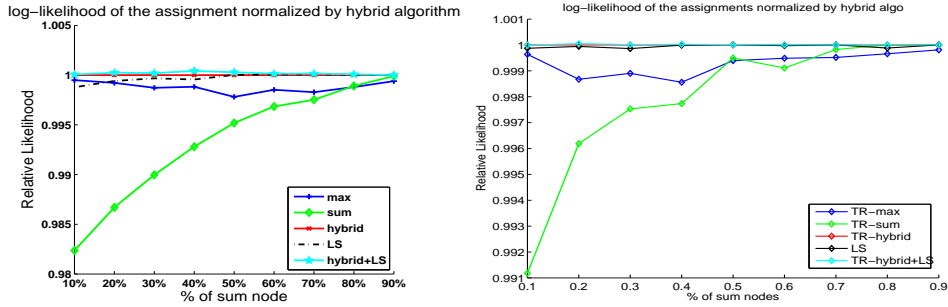

Figure 4: Approximate Log-Partition Function Scores on a 50-Node Tree (Left) and an 8*10 Grid (Right) Graph Normalized By the Result of Hybrid Algorithm

the local optimum, which often happened to be the global optimum as well. In particular, it only takes 1 or 2 steps of search in the 10-node chain case and 1 to 3 steps in the 50-node tree case.

Next, we experiment with the marginals estimation. Fig 2 shows the mean of KL-divergence on the marginals for the three message-passing algorithms (each averaged over 100 random experiments) compared to the true marginals of $p(z|x)$. Greedy search of [6] is not included since it only provides MAP, not marginals. The x-axis shows the percentage of sum nodes in the graph. Just like in the MAP case, our hybrid method consistently produces the smallest KL-divergence compared to others.

When the computation of the truth is intractable, the loglikelihood of the assignment can be approximated by the log-partition function with Bethe approximation according to Sec. 3.2. Note that this is exact on trees. Here, we use a 50-node tree with binary node states and $8 \times 10$ grid with various states $1 \leq |\mathcal{Y}_s| \leq 20$. On the grid graph, we apply tree-reweighted sum or max product [14, 13], and our hybrid version based on TRBP. For the edge appearance probability in TRBP, we apply a common approach that use a greedy algorithm to finding the spanning trees with as many uncovered edges as possible until all the edges in the graph are covered at least once. Even if the message-passing algorithms are not guaranteed to converge on loopy graphs, we can still compare the best result they provide after a certain number of iterations

Fig. 4 presents the results. In the tree case, as expected, using hybrid message-passing algorithms's result to initialize the local search algorithm performs the best. On the grid graph, the local search algorithm initialized by the sum product results works well when there are few max nodes, but as the search space grows exponentially with the number of max nodes, so it takes hundreds of steps to find the optimum. On the other hand, because the hybrid TRBP starts in a good area, it consistently achieves the highest likelihood among all four algorithms with fewer extra steps.

## 6.2 Real-world Data

We then experiment with the protein side-chain prediction dataset [23, 24] which consists a set of protein structures for which we need to find lowest energy assignment for rotamer residues. There are two sets of residues: core residues and surface residues. The core residues are the residues which are connected to more than 19 other residues, and the surface ones are the others. Since the MAP results are usually lower on the surface residues than the core residues nodes [24], we choose the surface residues to be max nodes and the core nodes to be the sum nodes. The ground truth is given by the maximum likelihood assignment of the residues, so we do not expect to have a better results on the core nodes, but

Table 1: Accuracy on the 1st, the 1st & 2rd Angles

| $\chi_1$ | ALL | SURFACE | CORE |
|---|---|---|---|
| sum product | 0.7900 | 0.7564 | 0.8325 |
| max product | 0.7900 | 0.7555 | 0.8336 |
| hybrid | 0.7910 | 0.7573 | 0.8336 |
| TRBP | 0.7942 | 0.7608 | **0.8364** |
| hybrid TRBP | **0.7950** | **0.7626** | 0.8359 |
| $\chi_1 \wedge \chi_2$ | ALL | SURFACE | CORE |
| sum product | 0.6482 | 0.6069 | 0.7005 |
| max product | 0.6512 | 0.6064 | 0.7078 |
| hybrid | 0.6485 | 0.6051 | 0.7033 |
| TRBP | 0.6592 | 0.6112 | 0.7174 |
| hybrid TRBP | **0.6597** | **0.6140** | **0.7186** |

we hope that any improvement on the accuracy of the surface nodes can make up the loss on the core nodes and thus give a better performance overall. As shown in Table 6.2, the improvements of the hybrid methods on the surface nodes are more than the loss the the core nodes, thus improving the overall performance.

## Footnotes

[1]Running the standard sum-product algorithm and choosing the maximum likelihood assignment for the max nodes is also called maximum marginal decoding [15, 16].

[2]This results in four different relaxations for different combinations of message types and the hybrid algorithm performed empirically the best.

[3]By directly applying Jensen's inequality to the objective function $\max_x \log \sum_z p(x, z)$

[4]The proofs are straightforward following Lemma 1 and 2 in [18] page 4-5. More details are in Appendix B of the supplementary material

[5]A detailed derivation is in Appendix B.4 of the supplementary material

# References

[1] Shankar Kumar, William Byrne, and Speech Processing. Minimum bayes-risk decoding for statistical machine translation. In *HLT-NAACL*, 2004.

[2] David Sontag and Tommi Jaakkola. New outer bounds on the marginal polytope. In *In Advances in Neural Information Processing Systems*, 2007.

[3] Amir Globerson and Tommi Jaakkola. Fixing max-product: Convergent message passing algorithms for map lp-relaxations. In *NIPS*, 2007.

[4] Pradeep Ravikumar, Alekh Agarwal, and Martin J. Wainwright. Message-passing for graph-structured linear programs: proximal projections, convergence and rounding schemes. In *ICML*, 2008.

[5] Qiang Liu and Alexander Ihler. Variational algorithms for marginal map. In *UAI*, 2011.

[6] James D. Park. MAP Complexity Results and Approximation Methods. In *UAI*, 2002.

[7] D. Koller and N. Friedman. *Probabilistic Graphical Models: Principles and Techniques*. MIT Press, 2009.

[8] Shaul K. Bar-Lev, Daoud Bshouty, Peter Enis, Gerard Letac, I-Li Lu, and Donald Richards. The diagonal multivariate natural exponential families and their classification. In *Journal of Theoretical Probability*, pages 883–929, 1994.

[9] Vaibhava Goel and William J. Byrne. Minimum Bayes-risk automatic speech recognition. *Computer Speech and Language*, 14(2), 2000.

[10] M. J. Wainwright and M. I. Jordan. Graphical Models, Exponential Families, and Variational Inference. *Foundations and Trends in Machine Learning*, 2008.

[11] Judea Pearl. *Probabilistic Reasoning in Intelligent Systems: Networks of Plausible Inference*. Morgan Kaufmann Publishers Inc., San Francisco, CA, USA, 1988.

[12] Jonathan S. Yedidia, William T. Freeman, and Yair Weiss. Generalized belief propagation. In *NIPS*, 2000.

[13] Martin J. Wainwright, Tommi S. Jaakkola, and Alan S. Willsky. Exact map estimates by tree agreement. In *NIPS*, 2002.

[14] Martin J. Wainwright, Tommi S. Jaakkola, and Alan S. Willsky. Tree-reweighted belief propagation algorithms and approximate ml estimation by pseudo-moment matching. In *AISTATS*, 2003.

[15] Mark Johnson. Why doesnt em find good hmm pos-taggers. In *EMNLP*, pages 296–305, 2007.

[16] Pradeep Ravikumar, Martin J. Wainwright, and Alekh Agarwal. Message-passing for graph-structured linear programs: Proximal methods and rounding schemes, 2008.

[17] A. P. Dempster, N. M. Laird, and D. B. Rubin. Maximum Likelihood from Incomplete Data via the EM algorithm. *Journal of The Royal Statistica Society*, 1977.

[18] Radford M. Neal and Geoffrey E. Hinton. A View of the EM Algorithm that Justifies Incremental, Sparse, and Other Variants. In *Learning in graphical models*, pages 355–368, 1999.

[19] Slav Petrov and Dan Klein. Discriminative log-linear grammars with latent variables. In *NIPS*, 2008.

[20] Ivan Titov and James Henderson. A latent variable model for generative dependency parsing. In *IWPT*, 2007.

[21] Ben Taskar, Vassil Chatalbashev, Daphne Koller, and Carlos Guestrin. Learning structured prediction models: a large margin approach. 2004.

[22] Ioannis Tsochantaridis, Google Inc, Thorsten Joachims, Thomas Hofmann, Yasemin Altun, and Yoram Singer. Large margin methods for structured and interdependent output variables. *Journal of Machine Learning Research*, 6:1453–1484, 2005.

[23] Chen Yanover, Talya Meltzer, and Yair Weiss. Linear programming relaxations and belief propagation – an empirical study. *Journal of Machine Learning Research*, 7:1887–1907, 2006.

[24] Chen Yanover, Ora Schueler-furman, and Yair Weiss. Minimizing and learning energy functions for side-chain prediction. In *RECOMB2007*, 2007.

